# Joint support recovery under high-dimensional scaling: Benefits and perils of $\ell_{1,\infty}$-regularization

**Sahand Negahban**
Department of Electrical Engineering and Computer Sciences
University of California, Berkeley
Berkeley, CA 94720-1770
sahand_n@eecs.berkeley.edu

**Martin J. Wainwright**
Department of Statistics, and Department of Electrical Engineering and Computer Sciences
University of California, Berkeley
Berkeley, CA 94720-1770
wainwrig@eecs.berkeley.edu

## Abstract

Given a collection of $r \geq 2$ linear regression problems in $p$ dimensions, suppose that the regression coefficients share partially common supports. This set-up suggests the use of $\ell_1/\ell_\infty$-regularized regression for joint estimation of the $p \times r$ matrix of regression coefficients. We analyze the high-dimensional scaling of $\ell_1/\ell_\infty$-regularized quadratic programming, considering both consistency rates in $\ell_\infty$-norm, and also how the minimal sample size $n$ required for performing variable selection grows as a function of the model dimension, sparsity, and overlap between the supports. We begin by establishing bounds on the $\ell_\infty$-error as well sufficient conditions for exact variable selection for fixed design matrices, as well as designs drawn randomly from general Gaussian matrices. These results show that the high-dimensional scaling of $\ell_1/\ell_\infty$-regularization is qualitatively similar to that of ordinary $\ell_1$-regularization. Our second set of results applies to design matrices drawn from standard Gaussian ensembles, for which we provide a sharp set of necessary and sufficient conditions: the $\ell_1/\ell_\infty$-regularized method undergoes a phase transition characterized by the rescaled sample size $\theta_{1,\infty}(n, p, s, \alpha) = n/\{(4 - 3\alpha)s \log(p - (2 - \alpha) s)\}$. More precisely, for any $\delta > 0$, the probability of successfully recovering both supports converges to 1 for scalings such that $\theta_{1,\infty} \geq 1 + \delta$, and converges to 0 for scalings for which $\theta_{1,\infty} \leq 1 - \delta$. An implication of this threshold is that use of $\ell_{1,\infty}$-regularization yields improved statistical efficiency if the overlap parameter is large enough ($\alpha > 2/3$), but performs worse than a naive Lasso-based approach for moderate to small overlap ($\alpha < 2/3$). We illustrate the close agreement between these theoretical predictions, and the actual behavior in simulations.

## 1 Introduction

The area of high-dimensional statistical inference is concerned with the behavior of models and algorithms in which the dimension $p$ is comparable to, or possibly even larger than the sample size $n$. In the absence of additional structure, it is well-known that many standard procedures—among them linear regression and principal component analysis—are not consistent unless the ratio $p/n$ converges to zero. Since this scaling precludes having $p$ comparable to or larger than $n$, an active line of research is based on imposing structural conditions on the data—for instance, sparsity, manifold constraints, or graphical model structure—and then studying conditions under which various polynomial-time methods are either consistent, or conversely inconsistent.

This paper deals with high-dimensional scaling in the context of solving multiple regression problems, where the regression vectors are assumed to have shared sparse structure. More specifically, suppose that we are given a collection of $r$ different linear regression models in $p$ dimensions, with regression vectors $\bar{\beta}^i \in \mathbb{R}^p$, for $i = 1, \ldots, r$. We let $S(\bar{\beta}^i) = \{j \mid \bar{\beta}^i_j \neq 0\}$ denote the support set of $\bar{\beta}^i$. In many applications—among them sparse approximation, graphical model selection, and image reconstruction—it is natural to impose a sparsity constraint, corresponding to restricting the cardinality $|S(\bar{\beta}^i)|$ of each support set. Moreover, one might expect some amount of overlap between the sets $S(\bar{\beta}^i)$ and $S(\bar{\beta}^j)$ for indices $i \neq j$ since they correspond to the sets of active regression coefficients in each problem. For instance, consider the problem of image denoising or reconstruction, using wavelets or some other type of multiresolution basis. It is well known that natural images tend to have sparse representations in such bases. Moreover, similar images—say the same scene taken from multiple cameras—would be expected to share a similar subset of active features in the reconstruction. Similarly, in analyzing the genetic underpinnings of a given disease, one might have results from different subjects and/or experiments, meaning that the covariate realizations and regression vectors would differ in their numerical values, but one expects the same subsets of genes to be active in controlling the disease, which translates to a condition of shared support in the regression coefficients. Given these structural conditions of shared sparsity in these and other applications, it is reasonable to consider how this common structure can be exploited so as to increase the statistical efficiency of estimation procedures.

In this paper, we study the high-dimensional scaling of block $\ell_1/\ell_\infty$ regularization. Our main contribution is to obtain some precise—and arguably surprising—insights into the benefits and dangers of using block $\ell_1/\ell_\infty$ regularization, as compared to simpler $\ell_1$-regularization (separate Lasso for each regression problem). We begin by providing a general set of sufficient conditions for consistent support recovery for both fixed design matrices, and random Gaussian design matrices. In addition to these basic consistency results, we then seek to characterize rates, for the particular case of standard Gaussian designs, in a manner precise enough to address the following questions.

(a) First, under what structural assumptions on the data does the use of $\ell_1/\ell_\infty$ block-regularization provide a quantifiable reduction in the scaling of the sample size $n$, as a function of the problem dimension $p$ and other structural parameters, required for consistency?

(b) Second, are there any settings in which $\ell_1/\ell_\infty$ block-regularization can be harmful relative to computationally less expensive procedures?

Answers to these questions yield useful insight into the *tradeoff* between computational and statistical efficiency. Indeed, the convex programs that arise from using block-regularization typically require a greater computational cost to solve. Accordingly, it is important to understand under what conditions this increased computational cost guarantees that fewer samples are required for achieving a fixed level of statistical accuracy.

As a representative instance of our theory, consider the special case of standard Gaussian design matrices and two regression problems ($r = 2$), with the supports $S(\bar{\beta}^1)$ and $S(\bar{\beta}^2)$ each of size $s$ and overlapping in a fraction $\alpha \in [0,1]$ of their entries. For this problem, we prove that block $\ell_1/\ell_\infty$ regularization undergoes a phase transition in terms of the rescaled sample size

$$\theta_{1,\infty}(n, p, s, \alpha) \quad := \quad \frac{n}{(4 - 3\alpha)s \log(p - (2 - \alpha)s)}. \tag{1}$$

In words, for any $\delta > 0$ and for scalings of the quadruple $(n, p, s, \alpha)$ such that $\theta_{1,\infty} \geq 1 + \delta$, the probability of successfully recovering both $S(\bar{\beta}^1)$ and $S(\bar{\beta}^2)$ converges to one, whereas for scalings such that $\theta_{1,\infty} \leq 1 - \delta$, the probability of success converges to zero. By comparison to previous theory on the behavior of the Lasso (ordinary $\ell_1$-regularized quadratic programming), the scaling (1) has two interesting implications. For the $s$-sparse regression problem with standard Gaussian designs, the Lasso has been shown [10] to undergo a phase transition as a function of the rescaled sample size

$$\theta_{\text{Las}}(n, p, s) \quad := \quad \frac{n}{2s \log(p - s)}, \tag{2}$$

so that solving two separate Lasso problems, one for each regression problem, would recover both supports for problem sequences $(n, p, s)$ such that $\theta_{\text{Las}} > 1$. Thus, one consequence of our analysis is to provide a precise confirmation of the natural intuition: if the data is well-aligned with the regularizer, then block-regularization increases statistical efficiency. On the other hand, our analysis also conveys a cautionary message: if the overlap is too small—more precisely, if $\alpha < 2/3$—then block $\ell_{1,\infty}$ is actually *harmful* relative to the naive Lasso-based approach. This fact illustrates that some care is required in the application of block regularization schemes.

The remainder of this paper is organized as follows. In Section 2, we provide a precise description of the problem. Section 3 is devoted to the statement of our main result, some discussion of its consequences, and illustration by comparison to empirical simulations.

## 2 Problem set-up

We begin by setting up the problem to be studied in this paper, including multivariate regression and family of block-regularized programs for estimating sparse vectors.

### 2.1 Multivariate regression

In this problem, we consider the following form of multivariate regression. For each $i = 1, \ldots, r$, let $\bar{\beta}^i \in \mathbb{R}^p$ be a regression vector, and consider the family of linear observation models

$$y^i = X^i \bar{\beta}^i + w^i, \qquad i = 1, 2, \ldots, r. \tag{3}$$

Here each $X^i \in \mathbb{R}^{n \times p}$ is a design matrix, possibly different for each vector $\bar{\beta}^i$, and $w^i \in \mathbb{R}^n$ is a noise vector. We assume that the noise vectors $w^i$ and $w^j$ are independent for different regression problems $i \neq j$. In this paper, we assume that each $w^i$ has a multivariate Gaussian $N(0, \sigma^2 I_{n \times n})$ distribution. However, we note that qualitatively similar results will hold for any noise distribution with sub-Gaussian tails (see the book [1] for more background).

### 2.2 Block-regularization schemes

For compactness in notation, we frequently use $\bar{B}$ to denote the $p \times r$ matrix with $\bar{\beta}^i \in \mathbb{R}^p$ as the $i^{th}$ column. Given a parameter $q \in [1, \infty]$, we define the $\ell_1/\ell_q$ block-norm as follows:

$$\|B\|_{\ell_1/\ell_q} := \sum_{k=1}^{p} \|(\beta_k^1, \beta_k^2, \ldots, \beta_k^r)\|_q, \tag{4}$$

corresponding to applying the $\ell_q$ norm to each row of $B$, and the $\ell_1$-norm across all of these blocks. We note that all of these block norms are special cases of the CAP family of penalties [12].

This family of block-regularizers (4) suggests a natural family of $M$-estimators for estimating $\bar{B}$, based on solving the block-$\ell_1/\ell_q$-regularized quadratic program

$$\widehat{B} \in \arg\min_{B \in \mathbb{R}^{p \times r}} \left\{ \frac{1}{2n} \sum_{i=1}^{r} \|y^i - X^i \beta^i\|_2^2 + \lambda_n \|B\|_{\ell_1/\ell_q} \right\}, \tag{5}$$

where $\lambda_n > 0$ is a user-defined regularization parameter. Note that the data term is separable across the different regression problems $i = 1, \ldots, r$, due to our assumption of independence on the noise vectors. Any coupling between the different regression problems is induced by the block-norm regularization.

In the special case of univariate regression ($r = 1$), the parameter $q$ plays no role, and the block-regularized scheme (6) reduces to the Lasso [7, 3]. If $q = 1$ and $r \geq 2$, the block-regularization function (like the data term) is separable across the different regression problems $i = 1, \ldots, r$, and so the scheme (6) reduces to solving $r$ separate Lasso problems. For $r \geq 2$ and $q = 2$, the program (6) is frequently referred to as the group Lasso [11, 6]. Another important case [9, 8], and the focus of this paper, is block $\ell_1/\ell_\infty$ regularization.

The motivation for using block $\ell_1/\ell_\infty$ regularization is to encourage *shared sparsity* among the columns of the regression matrix $B$. Geometrically, like the $\ell_1$ norm that underlies the ordinary Lasso, the $\ell_1/\ell_\infty$ block norm has a polyhedral unit ball. However, the block norm captures potential interactions between the columns $\beta^i$ in the matrix $B$. Intuitively, taking the maximum encourages the elements $(\beta_k^1, \beta_k^2 \ldots, \beta_k^r)$ in any given row $k = 1, \ldots, p$ to be zero simultaneously, or to both be non-zero simultaneously. Indeed, if $\beta_k^i \neq 0$ for at least one $i \in \{1, \ldots, r\}$, then there is no additional penalty to have $\beta_k^j \neq 0$ as well, as long as $|\beta_k^j| \leq |\beta_k^i|$.

### 2.3 Estimation in $\ell_\infty$ norm and support recovery

For a given $\lambda_n > 0$, suppose that we solve the block $\ell_1/\ell_\infty$ program, thereby obtaining an estimate

$$\widehat{B} \in \arg\min_{B \in \mathbb{R}^{p \times r}} \left\{ \frac{1}{2n} \sum_{i=1}^{r} \|y^i - X^i \beta^i\|_2^2 + \lambda_n \|B\|_{\ell_1/\ell_\infty} \right\}, \tag{6}$$

We note that under high-dimensional scaling ($p \gg n$), this convex program (6) is not necessarily strictly convex, since the quadratic term is rank deficient and the block $\ell_1/\ell_\infty$ norm is polyhedral, which implies that the program is not strictly convex. However, a consequence of our analysis is that under appropriate conditions, the optimal solution $\widehat{B}$ is in fact unique.

In this paper, we study the accuracy of the estimate $\widehat{B}$, as a function of the sample size $n$, regression dimensions $p$ and $r$, and the sparsity index $s = \max_{i=1,\ldots,r} |S(\bar{\beta}^i)|$. There are various metrics with which to assess the "closeness" of the estimate $\widehat{B}$ to the truth $\bar{B}$, including predictive risk, various types of norm-based bounds on the difference $\widehat{B} - \bar{B}$, and variable selection consistency. In this paper, we prove results bounding the $\ell_\infty/\ell_\infty$ difference

$$\|\widehat{B} - \bar{B}\|_{\ell_\infty/\ell_\infty} \quad := \quad \max_{k=1,\ldots,p} \ \max_{i=1,\ldots,r} |\widehat{B}_k^i - \bar{B}_k^i|.$$

In addition, we prove results on support recovery criteria. Recall that for each vector $\bar{\beta}^i \in \mathbb{R}^p$, we use $S(\bar{\beta}^i) = \{k \mid \bar{\beta}_k^i \neq 0\}$ to denote its support set. The problem of *union support recovery* corresponds to recovering the set

$$J \quad := \quad \bigcup_{i=1}^{r} S(\bar{\beta}^i), \tag{7}$$

corresponding to the subset $J \subseteq \{1,\ldots,p\}$ of indices that are active in at least one regression problem. Note that the cardinality of $|J|$ is upper bounded by $rs$, but can be substantially smaller (as small as $s$) if there is overlap among the different supports.

In some results, we also study the more refined criterion of recovering the *individual signed supports*, meaning the signed quantities $\operatorname{sign}(\bar{\beta}_k^i)$, where the sign function is given by

$$\operatorname{sign}(t) \quad = \quad \begin{cases} +1 & \text{if } t > 0 \\ 0 & \text{if } t = 0 \\ -1 & \text{if } t < 0 \end{cases} \tag{8}$$

There are multiple ways in which the support (or signed support) can be estimated, depending on whether we use primal or dual information from an optimal solution.

$\ell_1/\ell_\infty$ **primal recovery:** Solve the block-regularized program (6), thereby obtaining a (primal) optimal solution $\widehat{B} \in \mathbb{R}^{p \times r}$, and estimate the signed support vectors

$$[\mathbb{S}_{\mathrm{pri}}(\widehat{\beta}^i)]_k \quad = \quad \operatorname{sign}(\widehat{\beta}_k^i). \tag{9}$$

$\ell_1/\ell_\infty$ **dual recovery:** Solve the block-regularized program (6), thereby obtaining an primal solution $\widehat{B} \in \mathbb{R}^{p \times r}$. For each row $k = 1,\ldots,p$, compute the set $\mathbb{M}_k := \arg \max_{i=1,\ldots,r} |\widehat{\beta}_k^i|$. Estimate the signed support via:

$$[\mathbb{S}_{\mathrm{dua}}(\widehat{\beta}_k^i)] \quad = \quad \begin{cases} \operatorname{sign}(\widehat{\beta}_k^i) & \text{if } i \in \mathbb{M}_k \\ 0 & \text{otherwise.} \end{cases} \tag{10}$$

As our development will clarify, this procedure corresponds to estimating the signed support on the basis of a dual optimal solution associated with the optimal primal solution.

## 2.4  Notational conventions

Throughout this paper, we use the index $p \in \{1,\ldots,r\}$ as a superscript in indexing the different regression problems, or equivalently the columns of the matrix $\bar{B} \in \mathbb{R}^{p \times r}$. Given a design matrix $X \in \mathbb{R}^{n \times p}$ and a subset $S \subseteq \{1,\ldots,p\}$, we use $X_S$ to denote the $n \times |S|$ sub-matrix obtained by extracting those columns indexed by $S$. For a pair of matrices $A \in \mathbb{R}^{m \times \ell}$ and $B \in \mathbb{R}^{m \times n}$, we use the notation $\langle A, B \rangle := A^T B$ for the resulting $\ell \times n$ matrix.

We use the following standard asymptotic notation: for functions $f, g$, the notation $f(n) = \mathcal{O}(g(n))$ means that there exists a fixed constant $0 < C < +\infty$ such that $f(n) \leq Cg(n)$; the notation $f(n) = \Omega(g(n))$ means that $f(n) \geq Cg(n)$, and $f(n) = \Theta(g(n))$ means that $f(n) = \mathcal{O}(g(n))$ and $f(n) = \Omega(g(n))$.

# 3 Main results and their consequences

In this section, we provide precise statements of the main results of this paper. Our first main result (Theorem 1) provides sufficient conditions for deterministic design matrices $X^1, \ldots, X^r$. This result allows for an arbitrary number $r$ of regression problems. Not surprisingly, these results show that the high-dimensional scaling of block $\ell_1/\ell_\infty$ is *qualitatively similar* to that of ordinary $\ell_1$-regularization: for instance, in the case of random Gaussian designs and bounded $r$, our sufficient conditions in [5] ensure that $n = \Omega(s \log p)$ samples are sufficient to recover the union of supports correctly with high probability, which matches known results on the Lasso [10].

As discussed in the introduction, we are also interested in the more refined question: can we provide necessary and sufficient conditions that are sharp enough to reveal *quantitative differences* between ordinary $\ell_1$-regularization and block regularization? In order to provide precise answers to this question, our final two results concern the special case of $r = 2$ regression problems, both with supports of size $s$ that overlap in a fraction $\alpha$ of their entries, and with design matrices drawn randomly from the standard Gaussian ensemble. In this setting, our final two results (Theorem 2 and 3) show that block $\ell_1/\ell_\infty$ regularization undergoes a *phase transition* specified by the rescaled sample size. We then discuss some consequences of these results, and illustrate their sharpness with some simulation results.

## 3.1 Sufficient conditions for deterministic designs

In addition to the sample size $n$, problem dimensions $p$ and $r$, and sparsity index $s$, our results are stated in terms of the minimum eigenvalue $C_{\min}$ of the $|J| \times |J|$ matrices $\frac{1}{n}\langle X_J^i, X_J^i\rangle$—that is,

$$\lambda_{\min}\big(\frac{1}{n}\langle X_J^i, X_J^i\rangle\big) \geq C_{\min} \qquad \text{for all } i = 1, \ldots, r, \tag{11}$$

as well as an $\ell_\infty$-operator norm of their inverses:

$$\|\big(\frac{1}{n}\langle X_J^i, X_J^i\rangle\big)^{-1}\|_\infty \leq D_{\max} \qquad \text{for all } i = 1, \ldots, r. \tag{12}$$

It is natural to think of these quantites as being constants (independent of $p$ and $s$), although our results do allow them to scale.

We assume that the columns of each design matrix $X^i, i = 1, \ldots, r$ are normalized so that

$$\|X_k^i\|_2^2 \quad \leq \quad 2n \qquad \text{for all } k = 1, 2, \ldots p. \tag{13}$$

The choice of the factor 2 in this bound is for later technical convenience. We also require the following *incoherence condition* on the design matrix is satisified: there exists some $\gamma \in (0, 1]$ such that

$$\max_{\ell=1,\ldots,|J^c|} \sum_{i=1}^{r} \|\langle X_\ell^i, X_J^i(\langle X_J^i, X_J^i\rangle)^{-1}\rangle\|_1 \quad \leq \quad (1 - \gamma), \tag{14}$$

and we also define the *support minimum value* $\overline{B}_{\min} = \min_{k \in J} \max_{i=1,\ldots,r} |\bar{\beta}_k^i|$,

For a parameter $\xi > 1$ (to be chosen by the user), we define the probability

$$\phi_1(\xi, p, s) \quad := \quad 1 - 2\exp(-(\xi-1)[r + \log p]) - 2\exp(-(\xi^2 - 1)\log(rs)) \tag{15}$$

which specifies the precise rate with which the "high probability" statements in Theorem 1 hold.

**Theorem 1.** *Consider the observation model* (3) *with design matrices $X^i$ satisfying the column bound* (13) *and incoherence condition* (14). *Suppose that we solve the block-regularized $\ell_1/\ell_\infty$ convex program* (6) *with regularization parameter $\rho_n^2 \geq \frac{4\xi\sigma^2}{\gamma^2}\frac{r^2 + r\log(p)}{n}$ for some $\xi > 1$. Then with probability greater than $\phi_1(\xi, p, s) \to 1$, we are guaranteed that:*

  (a) *The block-regularized program has a unique solution $\widehat{B}$ such that $\bigcup_{i=1}^{r} S(\widehat{\beta}^i) \subseteq J$, and it satisfies the elementwise bound*

$$\max_{i=1,\ldots,r} \max_{k=1,\ldots,p} |\widehat{\beta}_k^i - \bar{\beta}_k^i| \quad \leq \quad \underbrace{\xi\sqrt{\frac{4\sigma^2}{C_{\min}}\frac{\log(rs)}{n}} + D_{\max}\,\rho_n}_{b_1(\xi, \rho_n, n, s)}. \tag{16}$$

*(b) If in addition $\bar{B}_{\min} \geq b_1(\xi, \rho_n, n, s)$, then $\bigcup_{i=1}^r S(\widehat{\beta}^i) = J$, so that the solution $\widehat{B}$ correctly specifies the union of supports $J$.*

**Remarks:** To clarify the scope of the claims, part (a) guarantees that the estimator recovers the union support $J$ correctly, whereas part (b) guarantees that for any given $i = 1, \ldots, r$ and $k \in S(\bar{\beta}^i)$, the sign $\text{sign}(\widehat{\beta}_k^i)$ is correct. Note that we are guaranteed that $\widehat{\beta}_k^i = 0$ for all $k \notin J$. However, *within* the union support $J$, when using primal recovery method, it is possible to have false non-zeros—i.e., there may be an index $k \in J \backslash S(\bar{\beta}^i)$ such that $\widehat{\beta}_k^i \neq 0$. Of course, this cannot occur if the support sets $S(\bar{\beta}^i)$ are all equal. This phenomenon is related to geometric properties of the block $\ell_1/\ell_\infty$ norm: in particular, for any given index $k$, when $\widehat{\beta}_k^j \neq 0$ for *some* $j \in \{1, \ldots, r\}$, then there is no further penalty to having $\widehat{\beta}_k^i \neq 0$ for other column indices $i \neq j$.

The dual signed support recovery method (10) is more conservative in estimating the individual support sets. In particular, for any given $i \in \{1, \ldots, r\}$, it only allows an index $k$ to enter the signed support estimate $\mathbb{S}_{\text{dua}}(\widehat{\beta}^i)$ when $|\widehat{\beta}_k^i|$ achieves the maximum magnitude (possibly non-unique) across all indices $i = 1, \ldots, r$. Consequently, Theorem 1 guarantees that the dual signed support method will never include an index in the individual supports. However, it may incorrectly exclude indices of some supports, but like the primal support estimator, it is always guaranteed to correctly recover the union of supports $J$.

We note that it is possible to ensure that under some conditions that the dual support method will correctly recover each of the individual signed supports, without any incorrect exclusions. However, as illustrated by Theorem 2, doing so requires additional assumptions on the size of the gap $|\bar{\beta}_k^i| - |\bar{\beta}_k^j|$ for indices $k \in B := S(\bar{\beta}^i) \cap S(\bar{\beta}^j)$.

## 3.2 Sharp results for standard Gaussian ensembles

Our results thus far show under standard mutual incoherence or irrepresentability conditions, the block $\ell_1/\ell_\infty$ method produces consistent estimators for $n = \Omega(s \log(p-s))$. In qualitative terms, these results match known scaling for the Lasso, or ordinary $\ell_1$-regularization. In order to provide keener insight into the (dis)advantages associated with using $\ell_1/\ell_\infty$ block regularization, we specialize the remainder of our analysis to the case of $r = 2$ regression problems, where the corresponding design matrices $X^i, i = 1, 2$ are sampled from the standard Gaussian ensemble [2, 4]—i.e., with i.i.d. rows $N(0, I_{p \times p})$. Our goal in studying this special case is to be able to make *quantiative comparisons* with the Lasso.

We consider a sequence of models indexed by the triplet $(p, s, \alpha)$, corresponding to the problem dimension $p$, support sizes $s$. and overlap parameter $\alpha \in [0, 1]$. We assume that $s \leq p/2$, capturing the intuition of a (relatively) sparse model. Suppose that for a given model, we take $n = n(p, s, \alpha)$ observations. according to equation (3). We can then study the probability of successful recovery as a function of the model triplet, and the sample size $n$.

In order to state our main result, we define the order parameter or rescaled sample size $\theta_{1,\infty}(n, p, s, \alpha) := \frac{n}{(4-3\alpha)s \log(p-(2-\alpha)s)}$. We also define the *support gap value* as well as $c_\infty$-gap $\bar{B}_{\text{gap}} = ||\bar{\beta}_B^1| - |\bar{\beta}_B^2||$, and $c_\infty = \frac{1}{\rho_n}\|T(\bar{B}_{\text{gap}})\|_\infty$, where $T(\bar{B}_{\text{gap}}) = \rho_n \wedge \bar{B}_{\text{gap}}$.

### 3.2.1 Sufficient conditions

We begin with a result that provides sufficient conditions for support recovery using block $\ell_1/\ell_\infty$ regularization.
**Theorem 2** (Achievability). *Given the observation model* (3) *with random design $X$ drawn with i.i.d. standard Gaussian entries, and consider problem sequences $(n, p, s, \alpha)$ for which $\theta_{1,\infty}(n, p, s, \alpha) > 1 + \delta$ for some $\delta > 0$. If we solve the block-regularized program* (6) *with $\rho_n = \xi\sqrt{\frac{\log p}{n}}$ and $c_\infty \to 0$, then with probability greater than $1 - c_1 \exp(-c_2 \log(p - (2 - \alpha)s))$, the following properties hold:*

*(i) The block $\ell_{1,\infty}$-program* (6) *has a unique solution $(\widehat{\beta}^1, \widehat{\beta}^2)$, with supports $S(\widehat{\beta}^1) \subseteq J$ and $S(\widehat{\beta}^2) \subseteq J$. Moreover, we have the elementwise bound*

$$\max_{i=1,2} \max_{k=1,\ldots,p} |\widehat{\beta}_k^i - \bar{\beta}_k^i| \leq \underbrace{\xi\sqrt{\frac{100\log(s)}{n}} + \rho_n\left[\frac{4s}{\sqrt{n}} + 1\right]}_{b_3(\xi, \rho_n, n, s)}, \tag{17}$$

(ii) *If the support minimum $\bar{B}_{\min} > 2b_3(\xi, \rho_n, n, s)$, then the primal support method successfully recovers the support union $J = S(\bar{\beta}^1) \cup S(\bar{\beta}^2)$. Moreover, using the primal signed support recovery method* (9), *we have*

$$[\mathbb{S}_{\mathrm{pri}}(\widehat{\beta}^i)]_k \;\; = \;\; \mathrm{sign}(\bar{\beta}_k^i) \qquad \textit{for all } k \in S(\bar{\beta}^i). \tag{18}$$

### 3.2.2 Necessary conditions

We now turn to the question of finding matching necessary conditions for support recovery.

**Theorem 3** (Lower bounds). *Given the observation model* (3) *with random design $X$ drawn with i.i.d. standard Gaussian entries.*

(a) *For problem sequences $(n, p, s, \alpha)$ such that $\theta_{1,\infty}(n, p, s, \alpha) < 1 - \delta$ for some $\delta > 0$ and for any non-increasing regularization sequence $\rho_n > 0$, no solution $\widehat{B} = (\widehat{\beta}^1, \widehat{\beta}^2)$ to the block-regularized program* (6) *has the correct support union $S(\widehat{\beta}^1) \cup S(\widehat{\beta}^2)$.*

(b) *Recalling the definition of $\bar{B}_{\mathrm{gap}}$, define the rescaled gap limit $c_2(\rho_n, \bar{B}_{\mathrm{gap}}) \; := \; \limsup_{(n,p,s)} \frac{\|T(\bar{B}_{\mathrm{gap}})\|_2}{\rho_n \sqrt{s}}$. If the sample size $n$ is bounded as*

$$n < (1 - \delta) \left[ (4 - 3\alpha) + (c_2(\rho_n, \bar{B}_{\mathrm{gap}}))^2 \right] s \log[p - (2 - \alpha)s]$$

*for some $\delta > 0$, then the dual recovery method* (10) *fails to recover the individual signed supports.*

It is important to note that $c_\infty \geq c_2$, which implies then that as long as $c_\infty \to 0$, then $c_2 \to 0$, so that the conditions of Theorem 3(a) and (b) are equivalent. However, note that if $c_2$ does not go to 0, then in fact, the method could fail to recover the correct support even if $\theta_{1,\infty} > 1 + \delta$. This result is key to understanding the $\ell_{1,\infty}$-regularization term. The gap between the vectors plays a fundamental role in in reducing the sampling complexity. Namely, if the gap is too large, then the sampling efficiency is greatly reduced as compared to if the gap is very small. In summary, while (a) and (b) seem equivalent on the surface, the requirement in (b) is in fact stronger than that in (a) and demonstrates the importance of condition (iii) in Theorem 2. It shows that if the gap is too large, then correct joint support recovery is not possible.

### 3.3 Illustrative simulations and some consequences

In this section, we provide some illustrative simulations that illustrate the phase transitions predicted by Theorems 2 and 3, and show that the theory provides an accurate description of practice even for relatively small problem sizes (e.g., $p = 128$). Figure 1 plots the probability of successful recovery of the individual signed supports using dual support recovery (10)—namely, $\mathbb{P}[\mathbb{S}_{\mathrm{dua}}(\widehat{\beta}^i) = \mathbb{S}_\pm(\bar{\beta}^i), \mathbb{S}_{\mathrm{dua}}(\widehat{\beta}^2) = \mathbb{S}_\pm(\bar{\beta}^2)]$ for $i = 1, 2$—versus the order parameter $\theta_{1,\infty}(n, p, s, \alpha)$. The plot contains four sets of "stacked" curves, each corresponding to a different choice of the overlap parameter, ranging from $\alpha = 1$ (left-most stack), to $\alpha = 0.1$ (right-most stack). Each stack contains three curves, corresponding to the problem sizes $p \in \{128, 256, 512\}$. In all cases, we fixed the support size $s = 0.1p$. The stacking behavior of these curves demonstrates that we have isolated the correct order parameter, and the step-function behavior is consistent with the theoretical predictions of a sharp threshold.

Theorems 2 and 3 have some interesting consequences, particularly in comparison to the behavior of the "naive" Lasso-based individual decoding of signed supports—that is, the method that simply applies the Lasso (ordinary $\ell_1$-regularization) to each column $i = 1, 2$ separately. By known results [10] on the Lasso, the performance of this naive approach is governed by the order parameter

$$\theta_{\mathrm{Las}}(n, p, s) \;\; = \;\; \frac{n}{2s \log(p - s)}, \tag{19}$$

meaning that for any $\delta > 0$, it succeeds for sequences such that $\theta_{\mathrm{Las}} > 1 + \delta$, and conversely fails for sequences such that $\theta_{\mathrm{Las}} < 1 - \delta$. To compare the two methods, we define the relative efficiency coefficient $R(\theta_{1,\infty}, \theta_{\mathrm{Las}})$ := $\theta_{\mathrm{Las}}(n, p, s)/\theta_{1,\infty}(n, p, s, \alpha)$. A value of $R < 1$ implies that the block method is more efficient, while $R > 1$ implies that the naive method is more efficient.

With this notation, we have the following:

**Corollary 1.** *The relative efficiency of the block $\ell_{1,\infty}$ program* (6) *compared to the Lasso is given by $R(\theta_{1,\infty}, \theta_{\mathrm{Las}}) = \frac{4 - 3\alpha}{2} \frac{\log(p - (2 - \alpha)s)}{\log(p - s)}$. Thus, for sublinear sparsity $s/p \to 0$, the block scheme has greater statistical efficiency for all overlaps $\alpha \in (2/3, 1]$, but lower statistical efficiency for overlaps $\alpha \in [0, 2/3)$.*

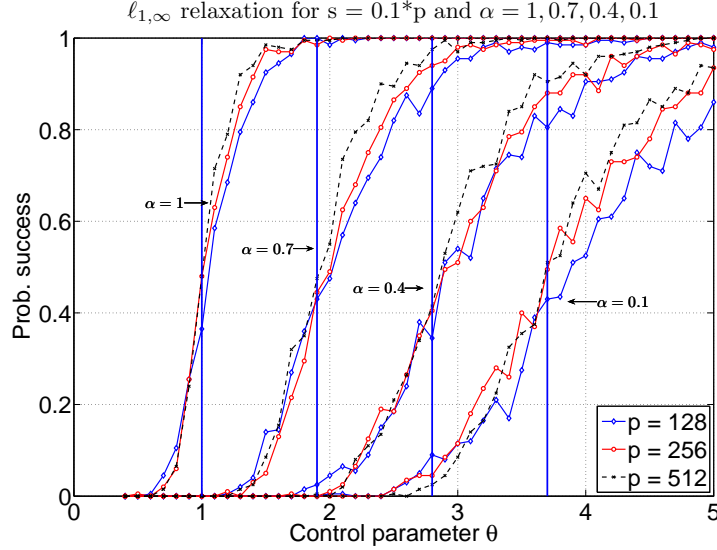

**Figure 1.** Probability of success in recovering the joint signed supports plotted against the control parameter $\theta_{1,\infty} = n/[2s\log(p - (2 - \alpha)s))]$ for linear sparsity $s = 0.1p$. Each stack of graphs corresponds to a fixed overlap $\alpha$, as labeled on the figure. The three curves within each stack correspond to problem sizes $p\{128, 256, 512\}$; note how they all align with each other and exhibit step-like behavior, consistent with Theorems 2 and 3. The vertical lines correspond to the thresholds $\theta_{1,\infty}^*(\alpha)$ predicted by Theorems 2 and 3; note the close agreement between theory and simulation.

# References

[1] V. V. Buldygin and Y. V. Kozachenko. *Metric characterization of random variables and random processes*. American Mathematical Society, Providence, RI, 2000.

[2] E. Candes and T. Tao. The Dantzig selector: Statistical estimation when $p$ is much larger than $n$. *Annals of Statistics*, 2006.

[3] S. Chen, D. L. Donoho, and M. A. Saunders. Atomic decomposition by basis pursuit. *SIAM J. Sci. Computing*, 20(1):33–61, 1998.

[4] D. L. Donoho and J. M. Tanner. Counting faces of randomly-projected polytopes when the projection radically lowers dimension. Technical report, Stanford University, 2006. Submitted to Journal of the AMS.

[5] S. Negahban and M. J. Wainwright. Joint support recovery under high-dimensional scaling: Benefits and perils of $\ell_{1,\infty}$-regularization. Technical report, Department of Statistics, UC Berkeley, January 2009.

[6] G. Obozinski, B. Taskar, and M. Jordan. Joint covariate selection for grouped classification. Technical report, Statistics Department, UC Berkeley, 2007.

[7] R. Tibshirani. Regression shrinkage and selection via the lasso. *Journal of the Royal Statistical Society, Series B*, 58(1):267–288, 1996.

[8] J. A. Tropp, A. C. Gilbert, and M. J. Strauss. Algorithms for simultaneous sparse approximation. *Signal Processing*, 86:572–602, April 2006. Special issue on "Sparse approximations in signal and image processing".

[9] B. Turlach, W.N. Venables, and S.J. Wright. Simultaneous variable selection. *Technometrics*, 27:349–363, 2005.

[10] M. J. Wainwright. Sharp thresholds for high-dimensional and noisy recovery of sparsity using using $\ell_1$-constrained quadratic programs. Technical Report 709, Department of Statistics, UC Berkeley, 2006.

[11] Kim Y., Kim J., and Y. Kim. Blockwise sparse regression. *Statistica Sinica*, 16(2), 2006.

[12] P. Zhao, G. Rocha, and B. Yu. Grouped and hierarchical model selection through composite absolute penalties. Technical report, Statistics Department, UC Berkeley, 2007.

